# NEURAL NET RECEIVERS IN MULTIPLE-ACCESS COMMUNICATIONS

Bernd-Peter Paris, Geoffrey Orsak, Mahesh Varanasi, Behnaam Aazhang
Department of Electrical and Computer Engineering
Rice University
Houston, TX 77251-1892

## ABSTRACT

The application of neural networks to the demodulation of spread-spectrum signals in a multiple-access environment is considered. This study is motivated in large part by the fact that, in a multiuser system, the conventional (matched filter) receiver suffers severe performance degradation as the relative powers of the interfering signals become large (the "near-far" problem). Furthermore, the optimum receiver, which alleviates the near-far problem, is too complex to be of practical use. Receivers based on multi-layer perceptrons are considered as a simple and robust alternative to the optimum solution. The optimum receiver is used to benchmark the performance of the neural net receiver; in particular, it is proven to be instrumental in identifying the decision regions of the neural networks. The back-propagation algorithm and a modified version of it are used to train the neural net. An importance sampling technique is introduced to reduce the number of simulations necessary to evaluate the performance of neural nets. In all examples considered the proposed neural net receiver significantly outperforms the conventional receiver.

## INTRODUCTION

In this paper we consider the problem of demodulating signals in a code-division multiple-access (CDMA) Gaussian channel. Multiple accessing in code domain is achieved by spreading the spectrum of the transmitted signals using preassigned code waveforms. The conventional method of demodulating a spread-spectrum signal in a multiuser environment employs one filter matched to the desired signal. Since the conventional receiver ignores the presence of interfering signals it is reliable only when there are few simultaneous transmissions. Furthermore, when the relative received power of the interfering signals become large (the "near-far" problem), severe performance degradation of the system is observed even in situations with relatively low bandwidth efficiencies (defined as the ratio of the number of channel subscribers to the spread of the bandwidth) [Aazhang 87]. For this reason there has been an interest in designing optimum receivers for multi-user communication systems [Verdu 86, Lupas 89, Poor 88]. The resulting optimum demodulators,

however, have a variable decoding delay with computational and storage complexity that depend exponentially on the number of active users. Unfortunately, this computational intensity is unacceptable in many applications. There is hence a need for near optimum receivers that are robust to near-far effects with a reasonable computational complexity to ensure their practical implementation.

In this study, we introduce a class of neural net receivers that are based on multilayer perceptrons trained via the back-propagation algorithm. Neural net receivers are very attractive alternatives to the optimum and conventional receivers due to their highly parallel structures. As we will observe, the performance of the neural net receivers closely track that of the optimum receiver in all examples considered.

## SYSTEM DESCRIPTION

In the multiple-access network of interest, transmitters are assumed to share a radio band in a combination of the time and code domain. One way of multiple accessing in the code domain is spread spectrum, which is a signaling scheme that uses a much wider bandwidth than necessary for a given data rate. Let us assume that in a given time interval there are $K$ active transmitters in the network. In a simple setting, the $k^{th}$ active user, in a symbol interval, transmits a signal from a binary signal set derived from the set of code waveforms assigned to the corresponding user. The signal is time limited to the interval $[0, T]$, where $T$ is the symbol duration.

In this paper we will concentrate on symbol-synchronous CDMA systems. Synchronous systems find applications in time slotted channels with the central (base) station transmitting to remote (mobile) terminals and also in relays between central stations. The synchronous problem will also be construed as providing us with a manageable setting to better understand the issues in the more difficult asynchronous situation. In a synchronous CDMA system, the users maintain time synchronism so that the relative time delays associated with all users are assumed to be zero. To illustrate the potentials of the proposed multiuser detector, we present the application to binary PSK direct-sequence signals in coherent systems. Therefore, the signal at a given receiver is the superposition of the $K$ transmitted signals in additive channel noise (see [Aazhang 87, Lupas 89] and references within)

$$r(t) = \sum_{i=1}^{P} \sum_{k=1}^{K} b_k^{(i)} A_k a_k(t - iT) \cos(\omega_c[t - iT] + \theta_k) + n_t, \quad t \in \Re, \qquad (1)$$

where $P$ is the packet length, $A_k$ is the signal amplitude, $\omega_c$ is the carrier frequency, $\theta_k$ is the phase angle. The symbol $b_k^{(i)} \in \{-1, +1\}$ denotes the bit that the $k^{th}$ user is transmitting in the $i^{th}$ time interval. In this model, $n_t$ is the additive channel noise which is assumed to be a white Gaussian random process. The time-limited code waveform, denoted by $a_k(t)$, is derived from the spreading sequence assigned to the $k^{th}$ user. That is, $a_k(t) = \sum_{j=0}^{N-1} a_j^{(k)} p(t - jT_c)$ where $p(t)$ is the unit rectangular pulse of duration $T_c$ and $N$ is the length of the spreading sequence. One code period $\underline{a}^{(k)} = [a_0^{(k)}, a_1^{(k)}, \ldots, a_{N-1}^{(k)}]$ is used for spreading the signal per symbol so

that $T = NT_c$. In this system, spectrum efficiency is measured as the ratio of the number of channel users to the spread factor, $K/N$.

In the next two sections, we first consider optimum synchronous demodulation of the multiuser spread-spectrum signal. Then, we introduce the application of neural networks to the multiuser detection problem.

# OPTIMUM RECEIVER

Multiuser detection is an active research area with the objective of developing strategies for demodulation of information sent by several transmitters sharing a channel [Verdu 86, Poor 88, Varanasi 89, Lupas 89]. In these situations with two or more users of a multiple-access Gaussian channel, one filter matched to the desired signal is no longer optimum since the decision statistics are effected by the other signals (e.g., the statistics are disturbed by cross-correlations with the interfering signals). Employing conventional matched filters, because of its structural simplicity, may still be justified if the system is operating at a low bandwidth efficiency. However, as the number of users in the system with fixed bandwidth grows or as the relative received powers of the interfering signals become large, severe performance degradation of the conventional matched filter is observed [Aazhang 87]. For direct-sequence spread-spectrum systems, optimum receivers obtained by Verdu and Poor require an extremely high degree of software complexity and storage, which may be unacceptable for most multiple-access systems [Verdu 86, Lupas 89]. Despite implementation problems, studies on optimum demodulation illustrate that the effects of interfering signals in a CDMA system, in principle, can be neutralized.

A complete study of the suboptimum neural net receiver requires a review of the maximum likelihood sequence detection formulation. Assuming that all possible information sequences are independent and equally likely, and defining $\underline{b}^{(i)} = [b_1^{(i)}, b_2^{(i)}, \ldots, b_K^{(i)}]'$, it is easy to see that an optimum decision on $\underline{b}^{(i)}$ is a one-shot decision in that it requires the observation of the received signal only in the $i^{th}$ time interval. Without loss of generality, we will therefore focus our attention on $i = 0$ and drop the time superscript and consider the demodulation of the vector of bits $\underline{b}$ with the observation of the received signal in the interval $[0, T]$.

In a $K$-user Gaussian channel, the most likely information vector is chosen as that which maximizes the log of the likelihood function (see [Lupas 89])

$$\underline{b}_{opt} = \arg \max_{\underline{b} \in \{-1, +1\}^K} \left\{ 2 \sum_k \int_0^T b_k S_k(t) r(t) dt - \int_0^T [\sum_k b_k S_k(t)]^2 dt \right\}, \quad (2)$$

where $S_k(t) = A_k a_k(t) \cos(\omega_c t + \theta_k)$ is the modulating signal of the $k^{th}$ user. The optimum decision can also be written as

$$\underline{b}_{opt} = \arg \max_{\underline{b} \in \{-1, +1\}^K} \left\{ 2\underline{y}'\underline{b} - \underline{b}'\mathbf{H}\underline{b} \right\}, \quad (3)$$

where $\mathbf{H}$ is the $K \times K$ matrix of signal cross-correlations such that the $(k, l)^{th}$ element is $h_{k,l} = < S_k(t), S_l(t) >$. The vector of sufficient statistics $\underline{y}$ consists of the

outputs of a bank of $K$ filters each matched to one of the signals

$$y_k = \int_0^T r(t)S_k(t)dt, \;\; for \;\; k = 1, 2, \ldots, K. \tag{4}$$

The maximization in (3) has been shown to be NP-complete [Lupas 89], i.e., no algorithm is known that can solve the maximization problem in polynomial time in $K$. This computational intensity is unacceptable in many applications. In the next section, we consider a suboptimum receiver that employs artificial neural networks for finding a solution to a maximization problem similar to (3).

# NEURAL NETWORK

Until now the application of neural networks to multiple-access communications has not drawn much attention. In this study we employ neural networks for classifying different signals in synchronous additive Gaussian channels. We assume that the information bits of the first of the $K$ signals is of interest, therefore, the phase angle of the desired signal is assumed to be zero (i.e., $\theta_1 = 0$). Two configurations with multi-layer perceptrons and sigmoid nonlinearity are considered for multiuser detection of direct-sequence spread-spectrum signals.

One structure is depicted in Figure 1.b where a layered network of perceptrons processes the sufficient statistics (4) of the multi-user Gaussian channel. In this structure the first layer of the net (referred to as the hidden layer) processes $[y_1, y_2, \ldots, y_K]$. The output layer may only have one node since there is only one signal that is being demodulated. This feed-forward structure is then trained using the back-propagation algorithm [Rumelhart 86].

In an alternate configuration, the continuous-time received signal is converted to an $N$-dimensional vector by sampling the output of the front-end filter at the chip rate $T_c^{-1}$ as illustrated in Figure 1.a. The input vector to the net can be written so that the demodulation of the first signal is viewed as a classification problem:

$$+ A_1' \underline{a}^{(1)} + \underline{\eta} + \underline{I} \;\; or \;\; - A_1' \underline{a}^{(1)} + \underline{\eta} + \underline{I}, \tag{5}$$

where $\underline{a}^{(1)}$ is the spreading code vector of the first user, $\underline{\eta}$ is a length-$N$ vector of filtered Gaussian noise samples and $\underline{I} = \sum_{k=2}^K b_k A_k' \cos(\theta_k)\underline{a}^{(k)}$ is the multiple-access interference vector with $A_k' = A_k T_c/2, \forall k = 1, 2, \ldots, K$. The layered neural net is then trained to process the input vector for demodulation of the first user's information bits via the back-propagation algorithm. For this configuration we consider two training methods, first the multi-layer receiver is trained, via the back-propagation algorithm, to classify the parity of the desired signal (referred to as the "trained" example) [Lippmann 87]. In another attempt (referred to as the "preset" example), the input layer of the net is preset as Gaussian classifiers and the other layers are trained using the back propagation algorithm [Gschwendtner 88].

Since we are interested in understanding the internal representation of knowledge by the weights of the net, a signal space method is developed to illustrate decision regions. In a $K$-user system where the spreading sequences are not orthogonal, the

signals can be represented by orthonormal bases using the Gram-Schmidt procedure. The optimum decision regions in the signal space for the demodulation of $b_1$ are known [Poor 88] and can be directly compared to ones for the neural net. Figure 2 illustrates decision regions for the optimum receiver and for "preset" and "trained" neural net receivers. In this example, two users are sharing a channel with $N = 3$, signal to noise ratio of user 1 ($SNR_1$) equal to $8dB$ and relative energies of the two user, $E_2/E_1 = 6dB$. As it is seen in this figure the decision region of the "preset" example is almost identical to the optimum boundary, however, the decision boundary for the "trained" example is quite conservative. Such comparisons are instrumental not only in identifying the pattern by which decisions are made by the neural networks but also in understanding the characteristics of the training algorithms.

# PERFORMANCE ANALYSIS

In this paper, we motivate the application of neural nets to single-user detection in multiuser channels by comparing the performance of the receivers in Figure 1 to that of the conventional and the optimum [Poor 88]. Since exact analysis of the bit error probabilities for the neural net receivers are analytically intractable, we consider Monte Carlo simulations. This method can produce very accurate estimates of bit-error probability if the number of simulations is sufficiently large to ensure occurrence of several erroneous decisions. The fact that these multiuser receivers operate with near optimum error rates puts a tremendous computational burden on the computer system. The new variance reduction scheme, developed by Orsak and Aazhang in [Orsak 89], first shifts the simulated channel noise to bias the simulations and then scales the error rate to obtain an unbiased estimate with a reduced variance. This importance sampling technique, which proved to be extremely effective in single-user detection [Orsak 89], is applied to the analysis of the multiuser systems.

As discussed in [Orsak 89], the fundamental issue is to generate more errors by biasing the simulations in cases where the error rate is very small. This strategy is better described by the two-user Gaussian example in Figure 2. In this example the simulation is carried out by generating zero-mean Gaussian noise vectors $\eta$, random phase $\theta_2$ and random values of the interfering bit $b_2$. Considering $b_1 = 1$ (corresponding to signals $+a_1 + a_2$ or $+a_1 - a_2$ which are marked by "+" in Figure 2) error occurs if the statistics fall on the left side of the decision boundary. It can be shown that the most efficient biasing scheme corresponds to a shift of the mean of the Gaussian noise and the multiple-access interference such that the mean of the statistics are placed on the decision boundary (the shifted signals are marked by "□" in Figure 2). Since this strategy generates much more errors than the standard Monte Carlo, errors are weighted to obtain an unbiased estimate of the error rate. The importance sampling technique substantially reduces the number of simulation trials compared to standard Monte Carlo for a given accuracy. In Figure 3 the gain which is defined as the ratio of the number of trials required for a fixed variance using Monte Carlo to that using the importance sampling method, is plotted versus

the bit-error probability. In this example, the spreading sequence length, $N$ is equal 3 and relative energies of the two user, $E_2/E_1 = 6dB$. The gain in this example of severe near-far problem is inversely proportional to the error rate. Furthermore, results from extensive analysis indicated that the proposed importance sampling technique is well suited for problems in multi-user communications and less than 100 trials is sufficient for an accurate error probability estimate.

# NUMERICAL RESULTS

The performance of the conventional, optimum [Poor 88] and the neural net receivers are compared via Monte Carlo simulations employing the importance sampling method. Except for a difference in length of training periods, the two configurations in Figure 1 result in similar average bit-error probabilities. Results presented here correspond to the neural net receiver in Figure 1.a.

A two-user Gaussian channel is considered with severe near-far problem where $E_2/E_1 = 6dB$ and spreading sequence length $N = 3$. In Figure 4, the average bit-error probabilities of the four receivers (conventional, optimum, neural nets for the "trained" and "preset" examples) are plotted versus the signal to noise ratio of the first user $(SNR_1)$. It is clear from this figure that the two neural net receivers outperform the matched filter receiver over the range of $SNR_1$. Figure 5 depicts these average error probabilities versus the relative energies of the two users (i.e., $E_2/E_1$ ) for a fixed $SNR_1 = 8dB$ and $N = 3$. As expected the conventional receiver becomes multiple-access limited as $E_2$ increases, however, the performance of the neural net receivers closely track that of the optimum receiver for all values of $E_2$.

We also considered a three-user Gaussian example with a high bandwidth efficiency and severe near-far problem where spreading sequence length $N = 3$ and first and third users have equal energy and second user has four times more energy (i.e., $E_2/E_1 = 6dB$ ). The average error probabilities of the four receivers versus $SNR_1$ are depicted in Figure 6. The neural net receivers maintained their near optimum performance even in this three user example with a spread factor of 3 corresponding to a bandwidth efficiency of 1.

# CONCLUSIONS

In this paper, we consider the problem of demodulating a signal in a multiple-access Gaussian channel. The error probability of different neural net receivers were compared with the conventional and optimum receivers in a symbol-synchronous system. As expected the performance of the conventional receiver (matched filter) is very sensitive to the strength of the interfering users. However, the error probability of the neural net receiver is independent of the strength of the other users and is at least one order of magnitude better than the conventional receiver. Except for a difference in the length of training periods, the two configurations in Figure 1 result in similar average bit-error probabilities. However, the training strategies, "preset" and "trained", resulted in slightly different error rates and decision regions.

The multi-layer perceptron was very successful in the classification problem in the presence of interfering signals. In all the examples that were considered, two layers

of perceptrons proved to be sufficient to closely approximate the decision boundary of the optimum receiver. We anticipate that this application of neural networks will shed more light on the potentials of neural nets in digital communications. The issues facing the project were quite general in nature and are reported in many neural network studies. However, we were able to address these issues in multiple-access communications since the disturbances are structured and the optimum receiver (which is NP-hard) is well understood.

# References

[Aazhang 87]    B. Aazhang and H. V. Poor. Performance of DS/SSMA Communications in Impulsive Channels-Part I: Linear Correlation Receivers. *IEEE Trans. Commun.*, COM-35(11):1179–1188, November 1987.

[Gschwendtner 88]    A. B. Gschwendtner. *DARPA Neural Network Study*. AFCEA International Press, 1988.

[Lippmann 87]    R. P. Lippmann and B. Gold. Neural-Net Classifiers Useful for Speech Recognition. In *IEEE First Conference on Neural Networks*, pages 417–425, San Diego, CA, June 21-24, 1987.

[Lupas 89]    R. Lupas and S. Verdu. Linear Multiuser Detectors for Synchronous Code-Division Multiple-Access Channels. *IEEE Trans. Info. Theory*, IT-34, 1989.

[Orsak 89]    G. Orsak and B. Aazhang. On the Theory of Importance Sampling Applied to the Analysis of Detection Systems. *IEEE Trans. Commun.*, COM-37, April, 1989.

[Poor 88]    H. V. Poor and S. Verdu. Single-User Detectors for Multiuser Channels. *IEEE Trans. Commun.*, COM-36(1):50–60, January, 1988.

[Rumelhart 86]    D. E. Rumelhart, G. E. Hinton, and R. J. Williams. Learning Internal Representation by Error Propagation. In D. E. Rumelhart and J. L. McClelland, editors, *Parallel Distributed Processing: Explorations in the Microstructure of Cognition. Vol. I: Foundations*, pages 318–362, MIT Press, 1986.

[Varanasi 89]    M. K. Varanasi and B. Aazhang. Multistage Detection in Asynchronous Code-Division Multiple-Access Communications. *IEEE Trans. Commun.*, COM-37, 1989.

[Verdu 86]    S. Verdu. Optimum Multiuser Asymptotic Efficiency. *IEEE Trans. Commun.*, COM-34(9):890–897, September, 1986.

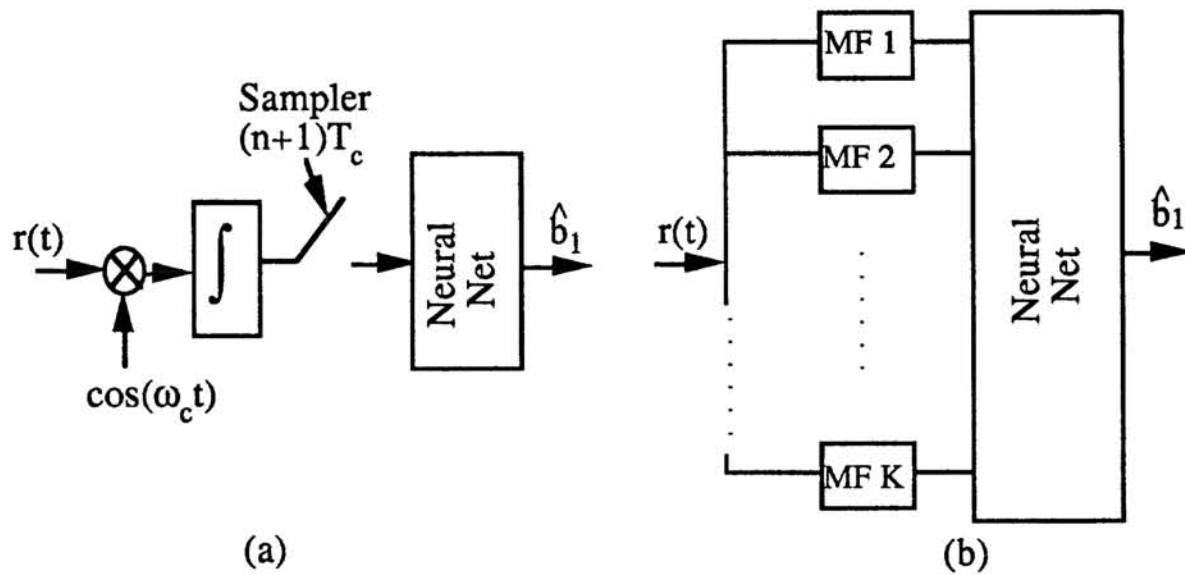

(a)                                                    (b)

**Figure 1. Two Neural Net Receiver Structures.**

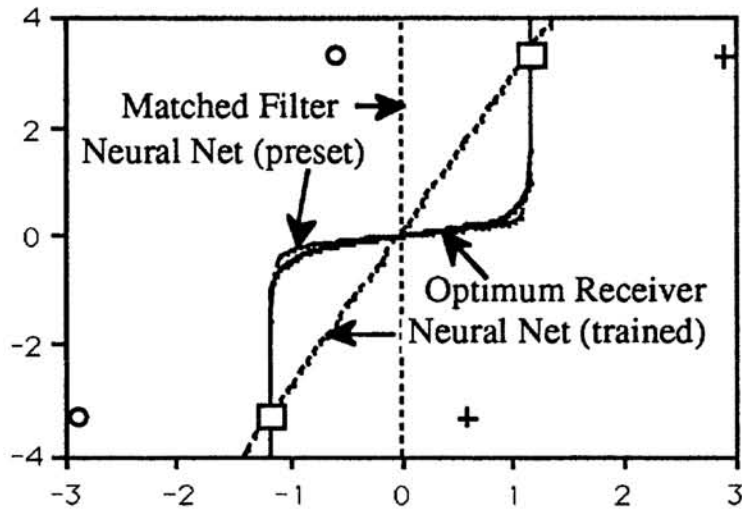

**Figure 2. Decision Boundaries of the Various Receivers.**

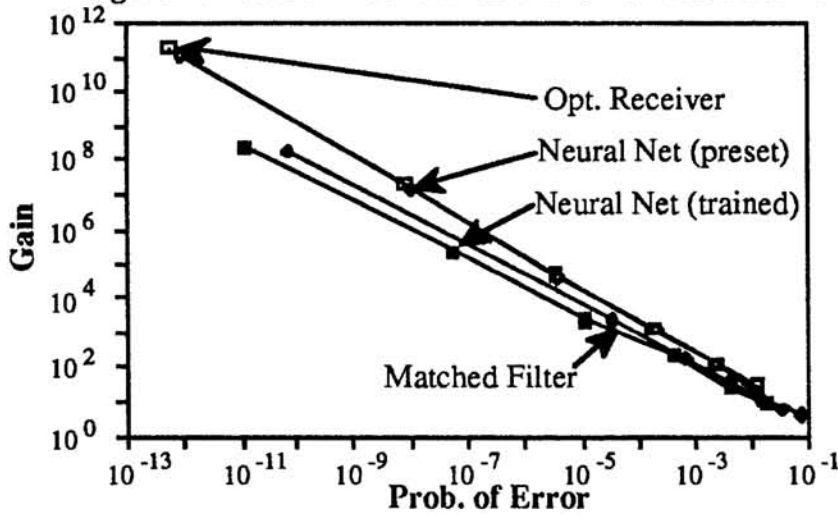

**Figure 3. Importance Sampling Gain versus Error Rate for 2-user Example.**

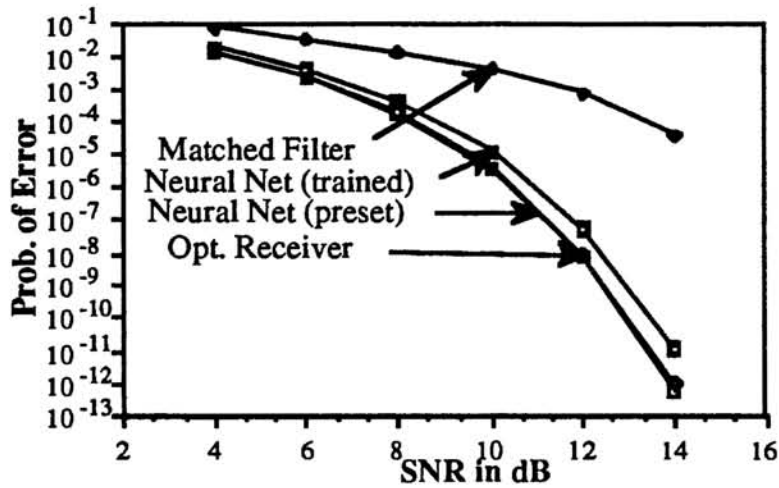

**Figure 4. Prob. of Error as a Function of the SNR (E2/E1 = 4).**

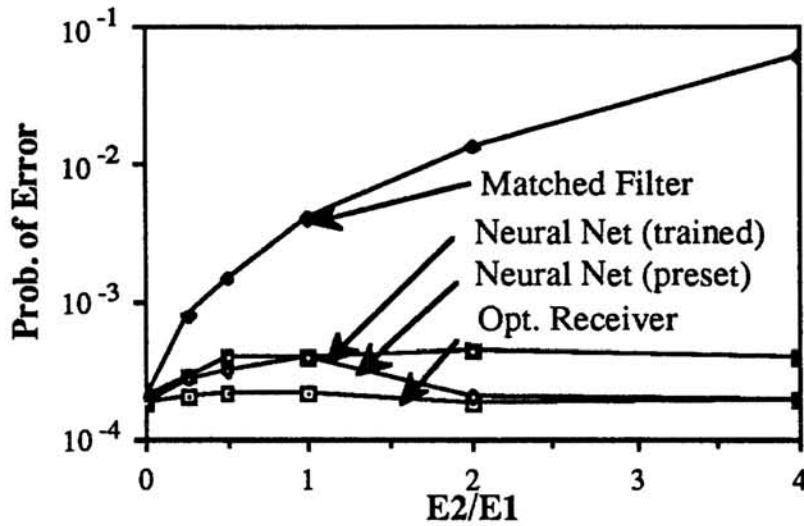

**Figure 5. Influence of MA-Interference (SNR = 8dB).**

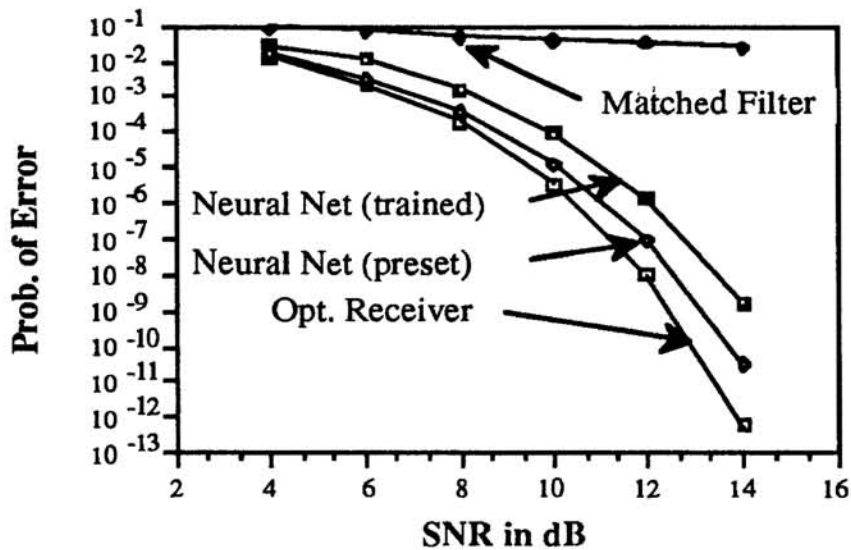

**Figure 6. Error Curves for the 3-User Example.**